# Laterally Interconnected Self-Organizing Maps in Hand-Written Digit Recognition

**Yoonsuck Choe, Joseph Sirosh, and Risto Miikkulainen**
Department of Computer Sciences
The University of Texas at Austin
Austin, TX 78712
yschoe,sirosh,risto@cs.utexas.edu

## Abstract

An application of laterally interconnected self-organizing maps (LISSOM) to handwritten digit recognition is presented. The lateral connections learn the correlations of activity between units on the map. The resulting excitatory connections focus the activity into local patches and the inhibitory connections decorrelate redundant activity on the map. The map thus forms internal representations that are easy to recognize with e.g. a perceptron network. The recognition rate on a subset of NIST database 3 is 4.0% higher with LISSOM than with a regular Self-Organizing Map (SOM) as the front end, and 15.8% higher than recognition of raw input bitmaps directly. These results form a promising starting point for building pattern recognition systems with a LISSOM map as a front end.

## 1   Introduction

Hand-written digit recognition has become one of the touchstone problems in neural networks recently. Large databases of training examples such as the NIST (National Institute of Standards and Technology) Special Database 3 have become available, and real-world applications with clear practical value, such as recognizing zip codes in letters, have emerged. Diverse architectures with varying learning rules have been proposed, including feed-forward networks (Denker et al. 1989; le Cun et al. 1990; Martin and Pittman 1990), self-organizing maps (Allinson et al. 1994), and dedicated approaches such as the neocognitron (Fukushima and Wake 1990).

The problem is difficult because handwriting varies a lot, some digits are easily confusable, and recognition must be based on small but crucial differences. For example, the digits 3 and 8, 4 and 9, and 1 and 7 have several overlapping segments, and the differences are often lost in the noise. Thus, hand-written digit recognition can be seen as a process of identifying the distinct features and producing an internal representation where the significant differences are magnified, making the recognition easier.

In this paper, the Laterally Interconnected Synergetically Self-Organizing Map architecture (LISSOM; Sirosh and Miikkulainen 1994, 1995, 1996) was employed to form such a separable representation. The lateral inhibitory connections of the LISSOM map *decorrelate* features in the input, retaining only those differences that are the most significant. Using LISSOM as a front end, the actual recognition can be performed by any standard neural network architecture, such as the perceptron.

The experiments showed that while direct recognition of the digit bitmaps with a simple perceptron network is successful 72.3% of the time, and recognizing them using a standard self-organizing map (SOM) as the front end 84.1% of the time, the recognition rate is 88.1% based on the LISSOM network. These results suggest that LISSOM can serve as an effective front end for real-world handwritten character recognition systems.

## 2   The Recognition System

### 2.1   Overall architecture

The system consists of two networks: a $20 \times 20$ LISSOM map performs the feature analysis and decorrelation of the input, and a single layer of 10 perceptrons the final recognition (Figure 1 $(a)$). The input digit is represented as a bitmap on the $32 \times 32$ input layer. Each LISSOM unit is fully connected to the input layer through the afferent connections, and to the other units in the map through lateral excitatory and inhibitory connections (Figure 1 $(b)$). The excitatory connections are short range, connecting only to the closest neighbors of the unit, but the inhibitory connections cover the whole map. The perceptron layer consists of 10 units, corresponding to digits 0 to 9. The perceptrons are fully connected to the LISSOM map, receiving the full activation pattern on the map as their input. The perceptron weights are learned through the delta rule, and the LISSOM afferent and lateral weights through Hebbian learning.

### 2.2   LISSOM Activity Generation and Weight Adaptation

The afferent and lateral weights in LISSOM are learned through Hebbian adaptation. A bitmap image is presented to the input layer, and the initial activity of the map is calculated as the weighted sum of the input. For unit $(i, j)$, the initial response $\eta_{ij}$ is

$$\eta_{ij} = \sigma \left( \sum_{a,b} \xi_{ab} \mu_{ij,ab} \right), \tag{1}$$

where $\xi_{ab}$ is the activation of input unit $(a, b)$, $\mu_{ij,ab}$ is the afferent weight connecting input unit $(a, b)$ to map unit $(i, j)$, and $\sigma$ is a piecewise linear approximation of the sigmoid activation function. The activity is then settled through the lateral connections. Each new activity $\eta_{ij}(t)$ at step $t$ depends on the afferent activation and the lateral excitation and inhibition:

$$\eta_{ij}(t) = \sigma \left( \sum_{a,b} \xi_{ab} \mu_{ij,ab} + \gamma_e \sum_{k,l} E_{ij,kl} \eta_{kl}(t-1) - \gamma_i \sum_{k,l} I_{ij,kl} \eta_{kl}(t-1) \right), \tag{2}$$

where $E_{ij,kl}$ and $I_{ij,kl}$ are the excitatory and inhibitory connection weights from map unit $(k, l)$ to $(i, j)$ and $\eta_{kl}(t-1)$ is the activation of unit $(k, l)$ during the previous time step. The constants $\gamma_e$ and $\gamma_i$ control the relative strength of the lateral excitation and inhibition.

After the activity has settled, the afferent and lateral weights are modified according to the Hebb rule. Afferent weights are normalized so that the length of the weight

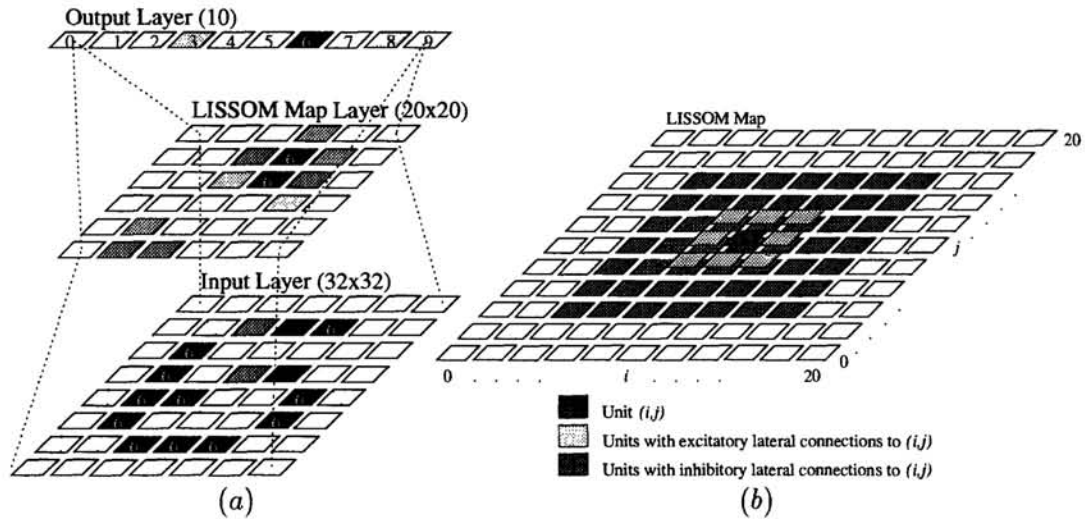

Figure 1: **The system architecture.** (a) The input layer is activated according to the bitmap image of digit 6. The activation propagates through the afferent connections to the LISSOM map, and settles through its lateral connections into a stable pattern. This pattern is the internal representation of the input that is then recognized by the perceptron layer. Through the connections from LISSOM to the perceptrons, the unit representing 6 is strongly activated, with weak activations on other units such as 3 and 8. (b) The lateral connections to unit $(i, j)$, indicated by the dark square, are shown. The neighborhood of excitatory connections (lightly shaded) is elevated from the map for a clearer view. The units in the excitatory region also have inhibitory lateral connections (indicated by medium shading) to the center unit. The excitatory radius is 1 and the inhibitory radius 3 in this case.

vector remains the same; lateral weights are normalized to keep the sum of weights constant (Sirosh and Miikkulainen 1994):

$$\mu_{ij,mn}(t+1) = \frac{\mu_{ij,mn}(t) + \alpha_{\text{inp}}\eta_{ij}\xi_{mn}}{\sqrt{\sum_{mn}[\mu_{ij,mn}(t) + \alpha_{\text{inp}}\eta_{ij}\xi_{mn}]^2}}, \tag{3}$$

$$\omega_{ij,kl}(t+1) = \frac{\omega_{ij,kl}(t) + \alpha\eta_{ij}\eta_{kl}}{\sum_{kl}[\omega_{ij,kl}(t) + \alpha\eta_{ij}\eta_{kl}]}, \tag{4}$$

where $\mu_{ij,mn}$ is the afferent weight from input unit $(m, n)$ to map unit $(i, j)$, and $\alpha_{\text{inp}}$ is the input learning rate; $\omega_{ij,kl}$ is the lateral weight (either excitatory $E_{ij,kl}$ or inhibitory $I_{ij,kl}$) from map unit $(k, l)$ to $(i, j)$, and $\alpha$ is the lateral learning rate (either $\alpha_{\text{exc}}$ or $\alpha_{\text{inh}}$).

### 2.3  Perceptron Output Generation and Weight Adaptation
The perceptrons at the output of the system receive the activation pattern on the LISSOM map as their input. The perceptrons are trained after the LISSOM map has been organized. The activation for the perceptron unit $O_m$ is

$$O_m = C \sum_{i,j} \eta_{ij}\nu_{ij,m}, \tag{5}$$

where $C$ is a scaling constant, $\eta_{ij}$ is the LISSOM map unit $(i, j)$, and $\nu_{ij,m}$ is the connection weight between LISSOM map unit $(i, j)$ and output layer unit $m$. The delta rule is used to train the perceptrons: the weight adaptation is proportional to the map activity and the difference between the output and the target:

$$\nu_{ij,m}(t+1) = \nu_{ij,m}(t) + \alpha_{\text{out}}\eta_{ij}(\zeta_m - O_m), \tag{6}$$

where $\alpha_{\text{out}}$ is the learning rate of the perceptron weights, $\eta_{ij}$ is the LISSOM map unit activity, $\zeta_m$ is the target activation for unit $m$. ($\zeta_m = 1$ if the correct digit $= m$, 0 otherwise).

| Representation | Training | Test |
|:---:|:---:|:---:|
| LISSOM | 93.0 / 0.76 | 88.1 / 3.10 |
| SOM | 84.5 / 0.68 | 84.1 / 1.71 |
| Raw Input | 99.2 / 0.06 | 72.3 / 5.06 |

Table 1: **Final Recognition Results.** The average recognition percentage and its variance over the 10 different splits are shown for the training and test sets. The differences in each set are statistically significant with $p > .9999$.

## 3 Experiments

A subset of 2992 patterns from the NIST Database 3 was used as training and testing data.[1] The patterns were normalized to make sure taht each example had an equal effect on the LISSOM map (Sirosh and Miikkulainen 1994). LISSOM was trained with 2000 patterns. Of these, 1700 were used to train the perceptron layer, and the remaining 300 were used as the validation set to determine when to stop training the perceptrons. The final recognition performance of the whole system was measured on the remaining 992 patterns, which neither LISSOM nor the perceptrons had seen during training. The experiment was repeated 10 times with different random splits of the 2992 input patterns into training, validation, and testing sets.

The LISSOM map can be organized starting from initially random weights. However, if the input dimensionality is large, as it is in case of the 32 × 32 bitmaps, each unit on the map is activated roughly to the same degree, and it is difficult to bootstrap the self-organizing process (Sirosh and Miikkulainen 1994, 1996). The standard Self-Organizing Map algorithm can be used to preorganize the map in this case. The SOM performs preliminary feature analysis of the input, and forms a coarse topological map of the input space. This map can then be used as the starting point for the LISSOM algorithm, which modifies the topological organization and learns lateral connections that decorrelate and represent a more clear categorization of the input patterns.

The initial self-organizing map was formed in 8 epochs over the training set, gradually reducing the neighborhood radius from 20 to 8. The lateral connections were then added to the system, and over another 30 epochs, the afferent and lateral weights of the map were adapted according to equations 3 and 4. In the beginning, the excitation radius was set to 8 and the inhibition radius to 20. The excitation radius was gradually decreased to 1 making the activity patterns more concentrated and causing the units to become more selective to particular types of input patterns. For comparison, the initial self-organized map was also trained for another 30 epochs, gradually decreasing the neighborhood size to 1 as well. The final afferent weights for the SOM and LISSOM maps are shown in figures 2 and 3.

After the SOM and LISSOM maps were organized, a complete set of activation patterns on the two maps were collected. These patterns then formed the training input for the perceptron layer. Two separate versions were each trained for 500 epochs, one with SOM and the other with LISSOM patterns. A third perceptron layer was trained directly with the input bitmaps as well.

Recognition performance was measured by counting how often the most highly active perceptron unit was the correct one. The results were averaged over the 10 different splits. On average, the final LISSOM+perceptron system correctly recognized 88.1% of the 992 pattern test sets. This is significantly better than the 84.1%

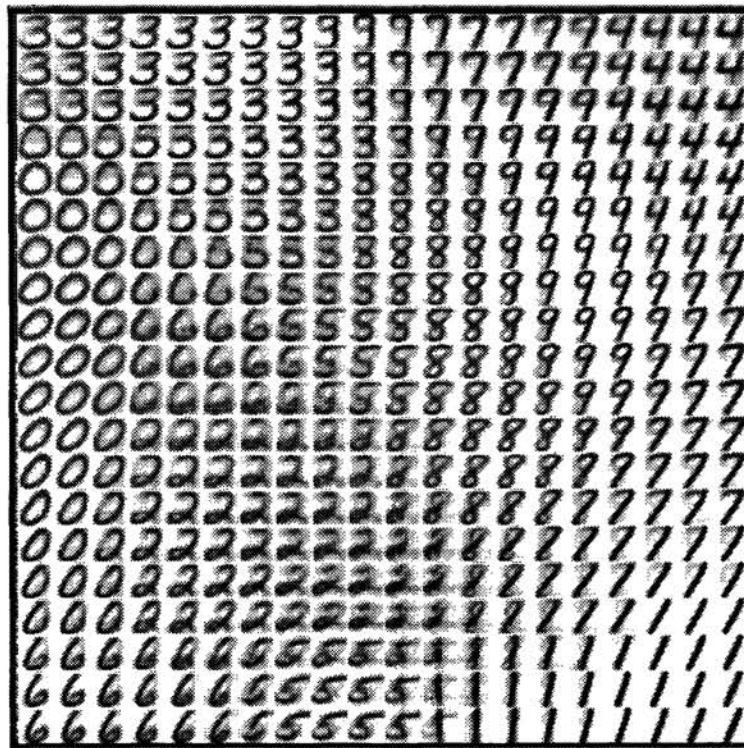

Figure 2: **Final Afferent Weights of the SOM map.** The digit-like patterns represent the afferent weights of each map unit projected on the input layer. For example, the lower left corner represents the afferent weights of unit $(0,0)$. High weight values are shown in black and low in white. The pattern of weights shows the input pattern to which this unit is most sensitive (6 in this case). There are local clusters sensitive to each digit category.

of the SOM+perceptron system, and the 72.3% achieved by the perceptron layer alone (Table 1). These results suggest that the internal representations generated by the LISSOM map are more distinct and easier to recognize than the raw input patterns and the representations generated by the SOM map.

## 4   Discussion

The architecture was motivated by the hypothesis that the lateral inhibitory connections of the LISSOM map would decorrelate and force the map activity patterns to become more distinct. The recognition could then be performed by even the simplest classification architectures, such as the perceptron. Indeed, the LISSOM representations were easier to recognize than the SOM patterns, which lends evidential support to the hypothesis. In additional experiments, the perceptron output layer was replaced by a two-weight-layer backpropagation network and a Hebbian associator net, and trained with the same patterns as the perceptrons. The recognition results were practically the same for the perceptron, backpropagation, and Hebbian output networks, indicating that the internal representations formed by the LISSOM map are the crucially important part of the recognition system.

A comparison of the learning curves reveals two interesting effects (figure 4). First, even though the perceptron net trained with the raw input patterns initially performs well on the test set, its generalization decreases dramatically during training. This is because the net only learns to memorize the training examples, which does not help much with new noisy patterns. Good internal representations are therefore crucial for generalization. Second, even though initially the settling process of the LISSOM map forms patterns that are significantly easier to recognize than

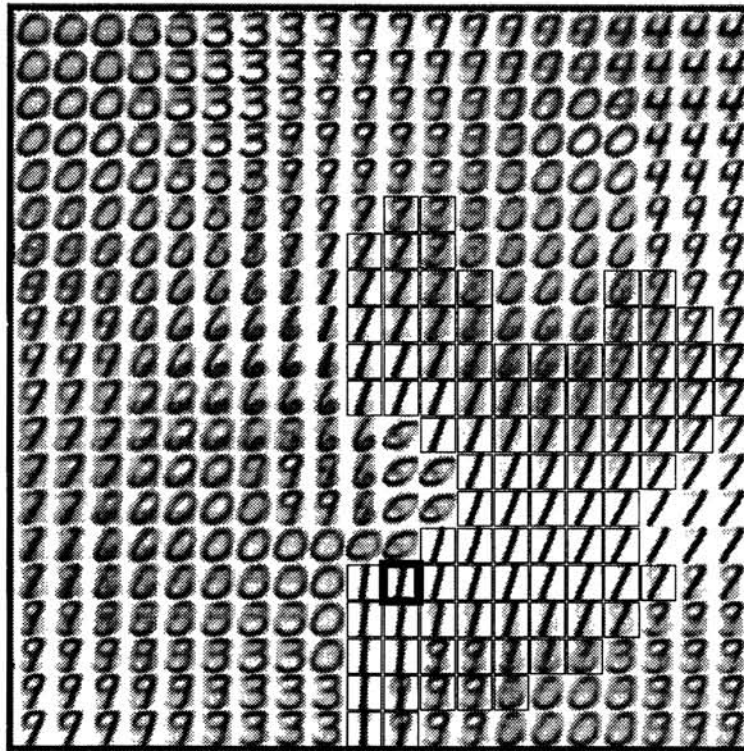

Figure 3: **Final Afferent Weights of the LISSOM map.** The squares identify the above-average inhibitory lateral connections to unit $(10, 4)$ (indicated by the thick square). Note that inhibition comes mostly from areas of similar functionality (i.e. areas sensitive to similar input), thereby decorrelating the map activity and forming a sparser representation of the input.

the initial, unsettled patterns (formed through the afferent connections only), this difference becomes insignificant later during training. The afferent connections are modified according to the final, settled patterns, and gradually learn to anticipate the decorrelated internal representations that the lateral connections form.

## 5   Conclusion

The experiments reported in this paper show that LISSOM forms internal representations of the input patterns that are easier to categorize than the raw inputs and the patterns on the SOM map, and suggest that LISSOM can form a useful front end for character recognition systems, and perhaps for other pattern recognition systems as well (such as speech). The main direction of future work is to apply the approach to larger data sets, including the full NIST 3 database, to use a more powerful recognition network instead of the perceptron, and to increase the map size to obtain a richer representation of the input space.

## Acknowledgements

This research was supported in part by National Science Foundation under grant #IRI-9309273. Computer time for the simulations was provided by the Pittsburgh Supercomputing Center under grants IRI930005P and IRI940004P, and by a High Performance Computer Time Grant from the University of Texas at Austin.

## Footnotes

[1]Downloadable at **ftp://sequoyah.ncsl.nist.gov/pub/databases/**.

## References

Allinson, N. M., Johnson, M. J., and Moon, K. J. (1994). Digital realisation of self-organising maps. In Touretzky, D. S., editor, *Advances in Neural Information Processing Systems 6*. San Mateo, CA: Morgan Kaufmann.

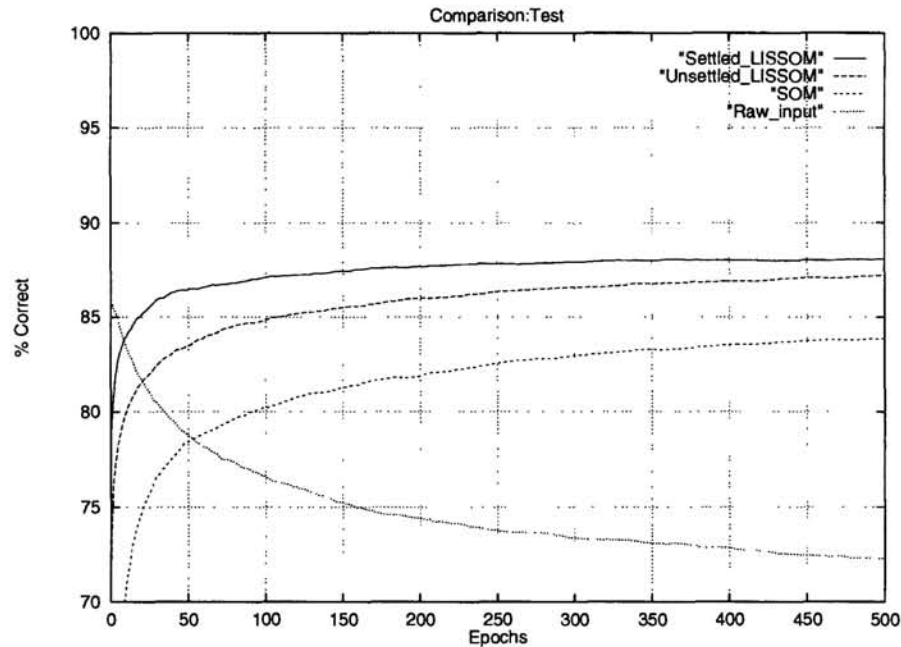

Figure 4: **Comparison of the learning curves.** A perceptron network was trained to recognize four different kinds of internal representations: the settled LISSOM patterns, the LISSOM patterns before settling, the patterns on the final SOM network, and raw input bitmaps. The recognition accuracy on the test set was then measured and averaged over 10 simulations. The generalization of the raw input + perceptron system decreases rapidly as the net learns to memorize the training patterns. The difference of using settled and unsettled LISSOM patterns diminishes as the afferent weights of LISSOM learn to take into account the decorrelation performed by the lateral weights.

Denker, J. S., Gardner, W. R., Graf, H. P., Henderson, D., Howard, R. E., Hubbard, W., Jackel, L. D., Baird, H. S., and Guyon, I. (1989). Neural network recognizer for hand-written zip code digits. In Touretzky, D. S., editor, *Advances in Neural Information Processing Systems 1*. San Mateo, CA: Morgan Kaufmann.

Fukushima, K., and Wake, N. (1990). Alphanumeric character recognition by neocognitron. In *Advanced Neural Computers*, 263–270. Elsevier Science Publishers B.V. (North-Holland).

le Cun, Y., Boser, B., Denker, J. S., Henderson, D., Howard, R. E., Hubbard, W., and Jackel, L. D. (1990). Handwritten digit recognition with a back-propagation network. In Touretzky, D. S., editor, *Advances in Neural Information Processing Systems 2*. San Mateo, CA: Morgan Kaufmann.

Martin, G. L., and Pittman, J. A. (1990). Recognizing hand-printed letters and digits. In Touretzky, D. S., editor, *Advances in Neural Information Processing Systems 2*. San Mateo, CA: Morgan Kaufmann.

Sirosh, J., and Miikkulainen, R. (1994). Cooperative self-organization of afferent and lateral connections in cortical maps. *Biological Cybernetics*, 71:66–78.

Sirosh, J., and Miikkulainen, R. (1995). Ocular dominance and patterned lateral connections in a self-organizing model of the primary visual cortex. In Tesauro, G., Touretzky, D. S., and Leen, T. K., editors, *Advances in Neural Information Processing Systems 7*. Cambridge, MA: MIT Press.

Sirosh, J., and Miikkulainen, R. (1996). Topographic receptive fields and patterned lateral interaction in a self-organizing model of the primary visual cortex. *Neural Computation* (in press).